# Learning Near-Pareto-Optimal Conventions in Polynomial Time

**Xiaofeng Wang**
ECE Department
Carnegie Mellon University
Pittsburgh, PA 15213
xiaofeng@andrew.cmu.edu

**Tuomas Sandholm**
CS Department
Carnegie Mellon University
Pittsburgh, PA 15213
sandholm@cs.cmu.edu

## Abstract

We study how to learn to play a Pareto-optimal strict Nash equilibrium when there exist multiple equilibria and agents may have different preferences among the equilibria. We focus on repeated coordination games of non-identical interest where agents do not know the game structure up front and receive noisy payoffs. We design efficient near-optimal algorithms for both the perfect monitoring and the imperfect monitoring setting(where the agents only observe their own payoffs and the joint actions).

## 1 Introduction

Recent years have witnessed a rapid development of multiagent learning theory. In particular, the use of reinforcement learning (RL) and game theory has attracted great attentions. However, research on multiagent RL (MARL) is still facing some rudimentary problems. Most importantly, *what is the goal of a MARL algorithm?* In a multiagent system, a learning agent generally cannot achieve its goal independent of other agents, which in turn tend to pursue their own goals. This questions the definition of optimality: No silver bullet guarantees maximization of each agent's payoff.

In the setting of self play (where all agents use the same algorithm), most existing MARL algorithms seek to learn to play a *Nash equilibrium*. It is the fixed point of the agents' best-response process, that is, each agent maximizes its payoff given the other's strategy. An equilibrium can be viewed as a *convention* that the learning agents reach for playing the unknown game. A key difficulty here is that a game usually contains multiple equilibria, and the agents need to coordinate on which one to play. Furthermore, the agents may have different preferences among the equilibria. Most prior work has avoided this problem by focusing on games with a unique equilibrium or games in which the agents have common interests.

In this paper, we advocate *Pareto-optimal Nash equilibria* as the equilibria that a MARL algorithm should drive agents to. This is a natural goal: Pareto-optimal equilibria are equilibria for which no other equilibrium exists where both agents are better off. We further design efficient algorithms for learning agents to achieve this goal in polynomial time.

## 2 Definitions and background

We study a repeated 2-agent game where the agents do not know the game up front, and try to learn how to play based on the experiences in the previous rounds of the game. As usual, we assume that the agents observe each others' actions. We allow for the possibility that the agents receive noisy but bounded payoffs (as is the case in many real-world MARL settings); this complicates the game because the joint action does not determine the agents' payoffs deterministically. Furthermore, the agents may prefer different outcomes of the game. In the next subsection we discuss the (stage) game that is repeated over and over.

### 2.1 Coordination games (of potentially non-identical interest)

We consider two agents, 1 and 2. The set of actions that agent $i$ can choose from is denoted by $A_i$. We denote the other agent by $-i$. Agents choose their individual actions $a_i \in A_i$ independently and concurrently. The results of their joint action can be represented in matrix form: The rows correspond to agent 1's actions and the columns correspond to agent 2's actions. Each cell $\{a_1, a_2\}$ in the matrix has the payoffs $u_1(\{a_1, a_2\}), u_2(\{a_1, a_2\})$. The agents may receive noisy payoffs. In this case, the $u_i$ functions are *expected* payoffs.

A *strategy* for agent $i$ is a distribution $\pi_i$ over its action set $A_i$. A *pure strategy* deterministically chooses one of the agent's individual actions. A *Nash equilibrium (NE)* is a strategy profile $\pi = \{\pi_i, \pi_{-i}\}$ in which no agent can improve its payoff by unilaterally deviating to a different strategy: $u_i(\{\pi_i, \pi_{-i}\}) \geq u_i(\{\pi_i', \pi_{-i}\})$ for both agents ($i = 1, 2$) and any strategy $\pi_i'$. We call a NE a *pure strategy NE* if the individuals' strategies in it are pure. Otherwise, we call it a *mixed strategy NE*. The NE is *strict* if we can replace "$\geq$" with "$>$".

We focus on the important and widely studied class of games called coordination games:[1]

**Definition 1 [Coordination game]** *A 2-agent coordination game G is an $N \times N$ matrix game with N strict Nash equilibria (called* conventions*). (It follows that there are no other pure-strategy equilibria.)*

A coordination game captures the notion that agents have the common interest of being coordinated (they both get higher payoffs by playing equilibria than other strategy profiles), but at the same time there are potentially non-identical interests (each agent may prefer different equilibria). The following small games illustrates this:

|  | OPT OUT | LARGE DEMAND | SMALL DEMAND |
|---|---|---|---|
| OPT OUT | 0,0 | 0,-0.1 | 0,-0.1 |
| SMALL DEMAND | -0.1,0 | 0.3,0.5 | 0.3,0.3 |
| LARGE DEMAND | -0.1,0 | -0.1,-0.1 | 0.5,0.3 |

Table 1: Two agents negotiate to split a coin. Each one can demand a small share (0.4) or a large share (0.6). There is a cost for bargaining (0.1). If the agents' demands add to less than 1, each one gets its demand. In this game, though agents favor different conventions, they would rather have a deal than opt out. The convention where both agents opt out is *Pareto-dominated* and the other two conventions are *Pareto-optimal*.

**Definition 2 [Pareto-optimality]** *A convention $\{a_1, a_2\}$ is Pareto-dominated if there exists at least one other convention $\{a_1', a_2'\}$ such that $u_i(\{a_1, a_2\}) < u_i(\{a_1', a_2'\})$ and $u_{-i}(\{a_1, a_2\}) \leq u_{-i}(\{a_1', a_2'\})$. If the inequality is strict, the Pareto domination is strict. Otherwise, it is weak. A convention is Pareto-optimal (PO) if and only if it is not Pareto-dominated.*

A Pareto-dominated convention is unpreferable because there is another convention that makes both agents better off. Therefore, we advocate that a MARL algorithm should at least cause agents to learn a PO convention.

In the rest of the paper we assume, without loss of generality, that the game is normalized so that all payoffs are strictly positive. We do this so that we can set artificial payoffs of zero (as described later) and be guaranteed that they are lower than any real payoffs. This is merely for ease of exposition; in reality we can set the artificial payoffs to a negative value below any real payoff.

## 2.2 Learning in game theory: Necessary background

Learning in game theory [6] studies repeated interactions of agents, usually with the goal of having the agents learn to play Nash equilibrium. There are key differences between learning in game theory and MARL. In the former, the agents are usually assumed to know the game before play, while in MARL the agents have to learn the game structure in addition to learning how to play. Second, the former has paid little attention to the efficiency of learning, a central issue in MARL. Despite the differences, the theory of learning in games has provided important principle for MARL.

One most widely used learning model is fictitious play *(FP)*. The basic FP does not guarantee to converge in coordination games while its variance, *adaptive play (AP)* [17], does. Therefore, we take AP as a building block for our MARL algorithms.

### 2.2.1 Adaptive play (AP)

The learning process of AP is as follows: Learning agents are assumed to have a memory to keep record of recent $m$ plays of the game. Let $a^t \in A$ be a joint action played at time $t$ over a game. Fix integers $k$ and $m$ such that $1 \leq k \leq m$. When $t \leq m$, each agent $i$ randomly chooses its actions. Starting from $t = m+1$, each agent looks back at the $m$ most recent plays $h_t = (a^{t-m}, a^{t-m+1}, \ldots, a^{t-1})$ and randomly (without replacement) selects $k$ samples from $h_t$. Let $K_t(a_{-i})$ be the number of times that an action $a_{-i} \in A_{-i}$ appears in the $k$ samples at $t$. Agent $i$ calculates its expected payoff w.r.t its individual action $a_i$ as $EP(a_i) = \sum_{a_{-i} \in A_{-i}} u_i(\{a_i, a_{-i}\}) \frac{K_t(a_{-i})}{k}$, and then randomly chooses an action from a set of best responses: $BR_i^t = \{a_i \mid a_i = \arg\max_{a_i' \in A_i} EP(a_i')\}$.

The learning process of AP can be modeled as a *Markov chain*. We take the initial *history* $h_m = (a^1, a^2, \ldots, a^m)$ as the *initial state* of the Markov chain. The definition of the other *states* is inductive: A successor of state $h$ is any state $h'$ obtained by deleting the left-most element of $h$ and appending a new right-most element. Let $h'$ be a successor of $h$, and let $a' = \{a_1', a_2'\}$ be the new element (joint action) that was appended to the right of $h$ to get $h'$. Let $p_{h,h'}$ be the transition probability from $h$ to $h'$. Now, $p_{h,h'} > 0$ if and only if for each agent $i$, there exists a sample of size $k$ in $h$ to which $a_i'$ is $i$'s best response. Because agent $i$ chooses such a sample with probability independent of time $t$, the Markov chain is stationary. In the Markov chain model, each state $h = (a, \ldots, a)$ with $a$ being a convention is an *absorbing state*. According to Theorem 1 in [17], AP in coordination games converge to such an absorbing state with probability 1 if $m \geq 4k$.

### 2.2.2 Adaptive play with persistent noise

AP does not choose a particular convention. However, Young showed that if there is *small constant noise* in action selection, AP usually selects a particular convention. Young studied the problem under an *independent random tremble* model: Suppose that instead of always taking a best-response action, with a small probability $\varepsilon$, the agent chooses a random action. This yields an irreducible and aperiodic perturbed process of the original Markov chain (unperturbed process). Young showed that with sufficiently small $\varepsilon$, the perturbed

process converges to a stationary distribution in which the probability to play so called *stochastic stable* convention(s) is at least $1 - C\varepsilon$, where $C$ is a positive constant (Theorem 4 and its proof in [17]).

The stochastic stable conventions of a game can be identified by considering the *mistakes* being made during state transitions. We say an agent made a mistake if it chose an action that is not a best response to any sample, of size $k$, taken from the $m$ most recent steps of history. Call the absorbing states in the unperturbed process *convention states* in the perturbed process. For each convention state $h$, we construct an *h-tree* $\tau_h$ (with each node being a convention state) such that there is a unique direct path from every other convention state to $h$. Label the direct edges $(v, v')$ in $\tau_h$ with the number of mistakes $r_{v,v'}$ needed to make the transition from convention state $v$ to convention state $v'$. The *resistance* of the $h$-tree is $r(\tau_h) = \sum_{(v,v') \in \tau_h} r_{v,v'}$. The *stochastic potential* of the convention state $h$ is the least resistance among all possible $h$-trees $\tau_h$. Young proved that the stochastic stable states are the states with the minimal stochastic potentials.

## 2.3 Reinforcement learning

*Reinforcement learning* offers an effective way for agents to estimate the expected payoffs associated with individual actions based on previous experience—without knowing the game structure. A simple and well-understood algorithm for single-agent RL is *Q-learning* [9]. The general form of Q-learning is for learning in a Markov decision process. It is more than we need here. In our single-state setting, we take a simplified form of the algorithm, with *Q-value* $Q_t^i(a)$ recording the estimate of the expected payoffs $u_i(a)$ for agent $i$ at time $t$. The agent updates its Q-values based on the sample of the payoff $R_t$ and the observed action $a$.

$$Q_{t+1}^i(a) = Q_t^i(a) + \alpha(R_t - Q_t^i(a)) \tag{1}$$

In single-agent RL, if each action is sampled infinitely and the learning rate $\alpha$ is decreased over time fast enough but not too fast, the Q-values will converge to agent $i$'s expected payoff $u_i$. In our setting, we set $\alpha = \frac{1}{\eta_t(a)}$, where $\eta_t(a)$ is the number of times that action $a$ has been taken.

Most early literature on RL was about asymptotic convergence to optimum. The extension of the convergence results to MARL include the minimax-Q [11], Nash-Q [8], friend-foe-Q [12] and correlated-Q [7]. Recently, significant attention has been paid to efficiency results: near-optimal polynomial-time learning algorithms. Important results include Fiechter's algorithm [5], Kearns and Singh's $E^3$ [10], Brafman and Tennenholtz's R-max [3], and Pivazyan and Shoham's efficient algorithms for learning a near-optimal policy [14]. These algorithms aim at efficiency, accumulating a provably close-to-optimal average payoff in polynomial running time with large probability. The equilibrium-selection problem in MARL has also been explored in the form of team games, a very restricted version of coordination games [4, 16].

In this paper, we develop efficient MARL algorithms for learning a PO convention in an unknown coordination game. We consider both the *perfect monitoring setting* where agents observe each others' payoffs, and the *imperfect monitoring setting* where agents do not observe each others' payoffs (and do not want to tell each other their payoffs). In the latter setting, our agents learn to play PO conventions without learning each others' preferences over conventions. Formally, the objectives of our MARL algorithms are:

**Efficiency:** Let $0 < \delta < 1$ and $\epsilon > 0$ be constants. Then with probability at least $1 - \delta$, agents will start to play a joint policy $a$ within steps polynomial in $\frac{1}{\epsilon}$, $\frac{1}{\delta}$, and $N$, such that there exists no convention $a'$ that satisfies $u_1(a) + \epsilon < u_1(a')$ and $u_2(a) + \epsilon < u_2(a')$. We call such a policy an $\epsilon$-*PO convention*.

# 3 An efficient algorithm for the perfect monitoring setting

In order to play an $\epsilon$-PO convention, agents need to find all these conventions first. Existing efficient algorithms employ random sampling to learn game $G$ before coordination. However, these approaches are thin for the goal: *Even when the game structure estimated from samples is within $\epsilon$ of $G$, its PO conventions might still be $\epsilon$ away from these of $G$.* Here we present a new algorithm to identify $\epsilon$-PO conventions efficiently.

**Learning game structure (perfect monitoring setting)**

1. Choose $\epsilon > 0, 0.5 > \delta > 0$. Set $w = 1$.
2. Compute the number of samples $M(\frac{\epsilon}{w}, \frac{\delta}{2^{w-1}})$ by using Chernoff/Hoeffding bound [14], such that $Pr\{\max_{a,i} |Q_M^i(a) - u_i(a)| \le \frac{\epsilon}{w}\} \ge 1 - \frac{\delta}{2^{w-1}}$.
3. Start from $t = 0$, randomly try $M$ actions with *uniform distributions* and update the Q-values using Equation 1.
4. If (1) $G_M = (Q_M^1, Q_M^2)$ has $N$ conventions, and (2) for every convention $\{a_i, a_{-i}\}$ in $G_M$ and every agent $i$, $Q_M^i(\{a_i, a_{-i}\}) > Q_M^i(\{a_i', a_{-i}\}) + 2\frac{\epsilon}{w}$ for every $a_i' \ne a_i$, then Stop; else $w \leftarrow w + 1$, Goto Step 2.

In Step 2 and Step 3, agent $i$ samples the coordination game $G$ sufficiently so that the game $G_M = (Q_M^1, Q_M^2)$ formed from $M$ samples is within $\frac{\epsilon}{w}$ of $G$ with probability at least $1 - \frac{\delta}{2^{w-1}}$. This is plausible because the agent can observe the other's payoffs. In Step 4, if Condition (1) and (2) are met and $G_M$ are within $\frac{\epsilon}{w}$ of $G$, we know that $G_M$ has the same set of conventions as $G$. So, any convention not strictly Pareto-dominated in $G_M$ is a $2\epsilon$-PO convention in $G$ by definition. The loop from Step 2 to Step 4 searches for a sufficiently small $\frac{\epsilon}{w}$ which has Condition (1) and (2) met. Throughout the learning, the probability that $G_M$ always stays within $\frac{\epsilon}{w}$ of $G$ after Step 3 is at least $1 - \sum_w \frac{\delta}{2^{w-1}} > 1 - 2\delta$. This implies that the algorithm will identify all the conventions of $G$ with probability at least $1 - 2\delta$. The total number of samples drawn is polynomial in $(N, \frac{1}{\delta}, \frac{1}{\epsilon})$ according to Chernoff bound [14].

After learning the game, the agents will further *learn how to play*, that is, to determine which PO convention in $G_M$ to choose. A simple solution is to let two agents randomize their action selection until they arrive at a PO convention in $G_M$. However, *this treatment is problematic because each agent may have different preferences over the conventions and thus will not randomly choose an action unless it believes the action is a best response to the other's strategy.* In this paper, we consider the learning agents which use adaptive play to negotiate the convention they should play. In game theory, AP was suggested as a simple learning model for bargaining [18], where each agent dynamically adjusts its offer w.r.t its belief about the other's strategy. Here we further propose a new algorithm called *k-step adaptive play (KSAP)* whose expected running time is polynomial in $m$ and $k$.

**Learning how to play (perfect monitoring setting)**

1. Let $VG_M = (Q_M^1, Q_M^2)$. Now, set those entries in $VG_M$ to zero that do not correspond to PO conventions.
2. Starting from a random initial state, sample the memory only every $k$ steps. Specifically, with probability 0.5, sample the most recent $k$ plays, otherwise, just randomly draw $k$ samples from the earlier $m - k$ observations without replacement.
3. Choose an action against $VG_M$ as in adaptive play *except that when there exist multiple best-response actions that correspond to some conventions in the game, choose an action that belongs to a convention that offers the greatest payoff (breaking remaining ties randomly).*
4. Play that action $k$ times.
5. Once observe that the last $k$ steps are composed of the same strict NE, play that NE forever.

In Step 1, agents construct a *virtual game* $VG_M$ from the game $G_M = (Q_M^1, Q_M^2)$ by setting the payoffs of all actions except PO conventions to zero. This eliminates all Pareto-dominated conventions in $G_M$. Step 2 to Step 5 is KSAP. Comparing with AP, KSAP lets an agent sample the experience to update its opponent model every $k$ steps. This makes the expected steps to reach an absorbing state polynomial in $k$. A KSAP agent pays more attentions on the most recent $k$ observations and will freeze its action once coordinated. This further enhances the performance of the learning algorithm.

**Theorem 1** *In any unknown 2-agent coordination game with perfect monitoring, if $m \ge 4k$, agents that use the above algorithm will learn a $2\epsilon$-PO policy with probability at least $1 - 2\delta$ in time $poly(N, \frac{1}{\delta}, \frac{1}{\epsilon}, m, k)$.*

Due to limited space, we present all proofs in a longer version of this paper [15].

## 4  An efficient algorithm for the imperfect monitoring setting

In this section, we present an efficient MARL algorithm for the imperfect monitoring setting where the agents do not observe each others' payoff during learning. Actually, since agents can observe joint actions, they may explicitly signal to each other their preferences over conventions through actions. This reduces the learning problem to that in the perfect monitoring setting. Here we assume that *agents are not willing to explicitly signal each other their preferences over conventions, even part of such information (e.g., their most preferable conventions).*[2] We study how to achieve optimal coordination without relying on such preference information.

Because each agent is unable to observe the other's payoffs and because there is noise in payoffs received, it is difficult for the agent to determine when enough samples have been taken to identify all conventions. We address this by allowing agents to demonstrate to each other their understanding of game structure (where the conventions are) after sampling.

**Learning the game structure (imperfect monitoring setting)**

1. Each agent plays its actions in order, with wrap around, until both agents have just wrapped around.[3] The agents name each others' actions 1,2,... according to the order of first appearance in play.
2. Given $\epsilon$ and $\delta$, agents are randomly sampling the game until every joint action has been visited at least $M(\frac{\epsilon}{w}, \frac{\delta}{2^w - 1})$ times (with $w = 1$) and updating their Q-values using Equation 1 along the way.
3. Starting at the same time, each agent $i$ goes through the other's $N$ individual actions $a_{-i}$ in order, playing the action $a_i$ such that $Q_M^i(\{a_i, a_{-i}\}) > 2\frac{\epsilon}{w} + Q_M^i(\{a_i', a_{-i}\})$ for any $a_i' \neq a_i$. (If such an action $a_i$ does not exists for some $a_{-i}$, then agent $i$ plays action 1 throughout this demonstration phase.)
4. Each agent determines whether the agents hold the same view of the $N$ strict Nash equilibria. If not, they let $w \leftarrow w + 1$, Goto Step 2.

After learning the game, the agents start to learn how to play. The difficulty is, without knowing about the other's preferences over conventions, agents cannot explicitly eliminate Pareto-dominated conventions in $G_M$. A straightforward approach is to allow each agent to choose its most preferable convention, and break tie randomly. This, however, requires to disclose the preference information to the other agent, thereby violating our assumption. Moreover, such a treatment limits the negotiation to only two solutions. Thus, even if there exists a better convention in which one agent compromise a little but the other is better off greatly, it will not be chosen. The intriguing question here is whether agents can learn to play a PO convention without knowing the other's preferences at all.

Adaptive play with persistent noise in action selection (see Section 2.2.2) causes agents to choose "stochastic stable" conventions most of time. This provides a potential solution to the above problem. Specifically, over $Q_M^i$, each agent $i$ first constructs a *best-response set* by including, for each possible action of the other agent $a_{-i}$, the joint action $\{a_i^*, a_{-i}\}$ where $a_i^*$ is $i$'s best response to $a_{-i}$. Then, agent $i$ forms a *virtual Q-function* $VQ_M^i$ which equals $Q_M^i$, except that the values of the joint actions not in the best-reponse set are zero. *We have proved that in the virtual game* $(VQ_M^1, VQ_M^2)$, *conventions strictly Pareto-dominated are not stochastic stable [15]. This implies that using AP with persistent noise, agents will play $2\epsilon$-PO conventions most of time even without knowing the other's preferences*. Therefore, if the agents can stop using noise in action selection at some point (and will thus play a particular convention from then on), there is a high probability that they end up playing a $2\epsilon$-PO convention. The rest of this section presents our algorithm in more detail.

We first adapt KSAP (see Section 3) to a learning model with persistent noise. After choosing the best-response action suggested by KSAP, each agent checks whether the current

state (containing the $m$ most recent joint actions) is a convention state. If it is not, the agent plays KSAP as usual (i.e., $k$ plays of the action selected). If it is, then in each of the following $k$ steps, the agent has probability $\varepsilon$ to randomly independently choose an action, and probability $1 - \varepsilon$ to play the best-response action. We call this algorithm $\varepsilon$-*KSAP*.

We can model this learning process as a Markov chain, with the state space including all and only convention states. Let $s_t$ be the state at time $t$ and $s_t^c$ be *the first convention state the agents reach after time* $t$. The transition probability is $p_{h,h'}^\varepsilon = Pr\{s_t^c = h'|s_t = h\}$, and it depends only on $h$, not $t$ (for a fixed $\varepsilon$). Therefore, the Markov chain is stationary. It is also irreducible and aperiodic, because with $\varepsilon > 0$, all actions have positive probability to be chosen in a convention state. Therefore, Theorem 4 in [17] applies and thus *the chain has a unique stationary distribution circling around the stochastic stable conventions of* $\{VQ^1, VQ^2\}$. These conventions are $2\epsilon$-PO (Lemma 5 in [15]) with probability $1 - 2\delta$. The proof of Lemma 1 in [17] further characterizes the support of the limit distribution. With $0 < \varepsilon < 1$, it is easy to obtain from the proof of Lemma 1 in [17] that the probability of playing $2\epsilon$-PO conventions is at least $1 - C\varepsilon$, where $C > 0$ is a constant.

Our algorithm intends to let agents stop taking noisy actions at some point and stick to a particular convention. This amounts to sampling the stationary distribution of the Markov chain. If the sampling is unbiased, the agents have a probability at least $1 - C\varepsilon$ to learn a $2\epsilon$-convention. The issue is how to make the sampling unbiased. We address this by applying a simple and efficient *Markov chain Monte Carlo algorithm* proposed by Lovász and Winkler [13]. The algorithm first randomly selects a state $h$ and randomly walks along the chain until all states have been visited. During the walk, it generates a function $A_h : S \setminus \{h\} \to S$, where $S$ is the set of all convention states. $A_h$ can be represented as a direct graph with a direct edge from each $h'$ to $A_h(h')$. After the walk, if agents find that $A_h$ defines an $h$-tree (see Section 2.2.2), $h$ becomes the convention the agents play forever. Otherwise, agents take another random sample from $S$ and repeat random walk, and so on. Lovász and Winkler proved that the algorithm makes an exact sampling of the Markov chain and that its expected running time is $O(\hbar^3 \log N)$, where $\hbar$ is the maximum expected time to transfer from one convention state to another. In our setting, we know that the probability to transit from one convention state to another is polynomial in $\varepsilon$ (probability to make mistakes in convention states). So, $\hbar$ is polynomial in $\frac{1}{\varepsilon}$. In addition, recall that our Markov chain is constructed on the convention states instead of all states. The expected time for making a transition in this chain is upper-bounded by the expected convergence time of KSAP which is polynomial in $m$ and $k$.

Recall that Lovász and Winkler's algorithm needs to do uniform random experiments when choosing $h$ and constructing $A_h$. In our setting, individual agents generate random numbers independently. Without knowing each others' random numbers, agents cannot commit to a convention together. If one of our learning agents commits to the final action before the other, the other may never commit because it is unable to complete the random walk. It is nontrivial to coordinate a joint commitment time between the agents because the agents cannot communicate (except via actions). We solve this problem by making the agents use the same random numbers (without requiring communication). We accomplish this via a *random hash function* technique, an idea common in cryptograhy [1]. Formally, a random hash function is a mapping from a pre-image space to an image space. Denote the random hash function with an image space $X$ by $\gamma_X$. It has two properties: (1) For any input, $\gamma_X$ randomly with uniform distribution draws an image from $X$ as an output. (2) With the same input, $\gamma_X$ gives the same output. Such functions are easy to construct (e.g., standard hash functions like MD5 and SHA can be converted to random hash functions by truncating their output [1]). In our learning setting, the agents share the same observations of previous plays. Therefore, we take the pre-image to be the most recent $m$ joint actions appended by the number of steps played so far. Our learning agents have the same random hash function $\gamma_X$. Whenever an agent should make a call to a random number generator, it

instead inputs to $\gamma_X$ the $m$ most recent joint actions and the total number of steps played so far, and uses the output of $\gamma_X$ as the random number. [4] This way the agents see the same uniform random numbers, and because the agents use the same algorithms, they will reach commitment to the final action at the same step.

**Learning how to play (imperfect monitoring setting)**

1. Construct a virtual Q-function $VQ^i$ from $Q_t^i$.
2. For $steps = 1, 2, 4, 8, \ldots$ do[5]
3. For $j = 1, 2, 3, \ldots, 3N$ do
4. $h = \gamma_S(h_t, t)$ (Use random hash function $\gamma_S$ to choose a convention state $h$ uniformly from $S$.)
5. $U = \{h\}$
6. Do until $U = S$
   (a) Play $\varepsilon$-KSAP until a convention state $h' \notin U$ is reached
   (b) $y = \gamma_{\{1,\ldots,steps\}}(h_{t'}, t')$
   (c) Play $\varepsilon$-KSAP until *convention states* have been visited $y$ times (counting duplicates). Denote the most recent convention state by $A_h(h')$
   (d) $U = U \cup \{h'\}$
7. If $A_h$ defines an $h$-tree, play $h$ forever
8. Endfor
9. Endfor

**Theorem 2** *In any unknown 2-agent coordination game with imperfect monitoring, for $0 < \varepsilon < 1$ and some constant $C > 0$, if $m \geq 4k$, using the above algorithm, the agents learn a $2\epsilon$-PO deterministic policy with probability at least $1 - 2\delta - C\varepsilon$ in time $poly(N, \frac{1}{\delta}, \frac{1}{\epsilon}, \frac{1}{\varepsilon}, m, k)$.*

## 5    Conclusions and future research

In this paper, we studied how to learn to play a Pareto-optimal strict Nash equilibrium when there exist multiple equilibria and agents may have different preferences among the equilibria. We focused on 2-agent repeated coordination games of non-identical interest where the agents do not know the game structure up front and receive noisy payoffs. We designed efficient near-optimal algorithms for both the perfect monitoring and the imperfect monitoring setting (where the agents only observe their own payoffs and the joint actions). In a longer version of the paper [15], we also present the convergence algorithms. In the future work, we plan to extend all these results to $n$-agent and multistage coordination games.

## Footnotes

[1] The term "coordination game" has sometimes been used to refer to special cases of coordination games, such as *identical-interest games* where agents have the same preferences [2], *minimum-effort games* that have strict Nash equilibria on the diagonal and both agents prefer equilibria further to the top left. Our definition is the most general (except that some have even called games that have weak Nash equilibria coordination games).

[2] Agents may prefer to hide such information to avoid giving others some advantage in the future interactions.

[3] In an $N \times N$ game this occurs for both agents at the same time, but the technique also works for games with a different number of actions per agent.

[4] Recall that agents have established the same numbering of actions. This allows them to encode their joint actions for inputting into $\gamma$ in the same way.

[5] The pattern of the for-loops is from the Lovász-Winkler algorithm [13].

## References

[1] Bellare and Rogaway. Random oracle are practical: A paradigm for designing efficient protocols. In *Proceedings of First ACM Annual Conference on Computer and Communication Security*, 93.
[2] Boutilier. Planning, learning and coordination in multi-agent decision processes. In *TARK*, 96.
[3] Brafman and Tennenholtz. R-max: A general polynomial time algorithm for near-optimal reinforcement learning. In *IJCAI*, 01.
[4] Claus and Boutilier. The dynamics of reinforcement learning in cooperative multi-agent systems. In *AAAI*, 98.
[5] Fiechter. Efficient reinforcement learning. In *COLT*, 94.
[6] Fudenberg and Levine. *The theory of learning in games*. MIT Press, 98.
[7] Greenwald and Hall. Correlated-q learning. In *AAAI Spring Symposium*, 02.
[8] Hu and Wellman. Multiagent reinforcement learning: theoretical framework and an algorithm. In *ICML*, 98.
[9] Kaelbling, Littman, and Moore. Reinforcement learning: A survey. *JAIR*, 96.
[10] Kearns and Singh. Near-optimal reinforcement learning in polynomial time. In *ICML*, 98.
[11] Littman. Value-function reinforcement learning in markov games. *J. of Cognitive System Research*, 2:55–66, 00.
[12] Littman. Friend-or-Foe Q-learning in general sum game. In *ICML*, 01.
[13] Lovász and Winkler. Exact mixing in an unknown markov chain. *Electronic Journal of Combinatorics*, 95.
[14] Pivazyan and Shoham. Polynomial-time reinforcement learning of near-optimal policies. In *AAAI*, 02.
[15] Wang and Sandholm. Learning to play pareto-optimal equilibria: Convergence and efficiency. www.cs.cmu.edu/~xiaofeng/LearnPOC.ps.
[16] Wang and Sandholm. Reinforcement learning to play an optimal Nash equilibrium in team markov game. In *NIPS*, 02.
[17] Young. The evolution of conventions. *Econometrica*, 61:57–84, 93.
[18] Young. An evolutionary model of bargaining. *Journal of Economic Theory*, 59, 93.

